# Estimating Conditional Probability Densities for Periodic Variables

**Chris M Bishop and Claire Legleye**
Neural Computing Research Group
Department of Computer Science and Applied Mathematics
Aston University
Birmingham, B4 7ET, U.K.
c.m.bishop@aston.ac.uk

## Abstract

Most of the common techniques for estimating conditional probability densities are inappropriate for applications involving periodic variables. In this paper we introduce three novel techniques for tackling such problems, and investigate their performance using synthetic data. We then apply these techniques to the problem of extracting the distribution of wind vector directions from radar scatterometer data gathered by a remote-sensing satellite.

## 1 INTRODUCTION

Many applications of neural networks can be formulated in terms of a multi-variate non-linear mapping from an input vector $\mathbf{x}$ to a target vector $\mathbf{t}$. A conventional neural network approach, based on least squares for example, leads to a network mapping which approximates the regression of $\mathbf{t}$ on $\mathbf{x}$. A more complete description of the data can be obtained by estimating the conditional probability density of $\mathbf{t}$, conditioned on $\mathbf{x}$, which we write as $p(\mathbf{t}|\mathbf{x})$. Various techniques exist for modelling such densities when the target variables live in a Euclidean space. However, a number of potential applications involve angle-like output variables which are periodic on some finite interval (usually chosen to be $(0, 2\pi)$). For example, in Section 3

we consider the problem of determining the wind direction (a periodic quantity) from radar scatterometer data obtained from remote sensing measurements. Most of the existing techniques for conditional density estimation cannot be applied in such cases.

A common technique for *unconditional* density estimation is based on mixture models of the form

$$p(\mathbf{t}) = \sum_{i=1}^{m} \alpha_i \phi_i(\mathbf{t}) \tag{1}$$

where $\alpha_i$ are called mixing coefficients, and the kernel functions $\phi_i(\mathbf{t})$ are frequently chosen to be Gaussians. Such models can be used as the basis of techniques for *conditional* density estimation by allowing the mixing coefficients, and any parameters governing the kernel functions, to be general functions of the input vector $\mathbf{x}$. This can be achieved by relating these quantities to the outputs of a neural network which takes $\mathbf{x}$ as input, as shown in Figure 1. Such an approach forms the basis of

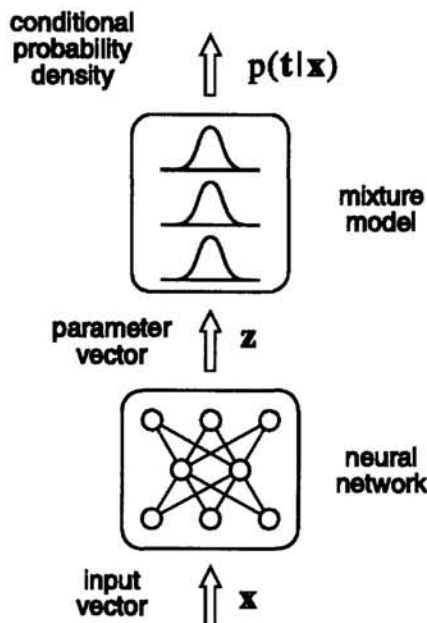

Figure 1: A general framework for conditional density estimation is obtained by using a feed-forward neural network whose outputs determine the parameters in a mixture density model. The mixture model then represents the conditional probability density of the target variables, conditioned on the input vector to the network.

the 'mixture of experts' model (Jacobs *et al.*, 1991) and has also been considered by a number of other authors (White, 1992; Bishop, 1994; Lui, 1994). In this paper we introduce three techniques for estimating conditional densities of *periodic* variables, based on extensions of the above formalism for Euclidean variables.

## 2 DENSITY ESTIMATION FOR PERIODIC VARIABLES

In this section we consider three alternative approaches to estimating the conditional density $p(\theta|\mathbf{x})$ of a periodic variable $\theta$, conditioned on an input vector $\mathbf{x}$. They are based respectively on a transformation to an extended domain representation, the use of adaptive circular normal kernel functions, and the use of fixed circular normal kernels.

### 2.1 TRANSFORMATION TO AN EXTENDED VARIABLE DOMAIN

The first technique which we consider involves finding a transformation from the periodic variable $\theta \in (0, 2\pi)$ to a Euclidean variable $\chi \in (-\infty, \infty)$, such that standard techniques for conditional density estimation can be applied in $\chi$-space. In particular, we seek a conditional density function $\widetilde{p}(\chi|\mathbf{x})$ which is to be modelled using a conventional Gaussian mixture approach as described in Section 1. Consider the transformation

$$p(\theta|\mathbf{x}) = \sum_{L=-\infty}^{\infty} \widetilde{p}(\theta + L2\pi|\mathbf{x}) \tag{2}$$

Then it is clear by construction that the density model on the left hand side satisfies the periodicity requirement $p(\theta + 2\pi|\mathbf{x}) = p(\theta|\mathbf{x})$. Furthermore, if the density function $\widetilde{p}(\chi|\mathbf{x})$ is normalized, then we have

$$
\begin{aligned}
\int_0^{2\pi} p(\theta|\mathbf{x})\, d\theta &= \sum_{L=-\infty}^{\infty} \int_0^{2\pi} \widetilde{p}(\theta + L2\pi|\mathbf{x})\, d\theta \\
&= \sum_{L=-\infty}^{\infty} \int_{L2\pi}^{(L+1)2\pi} \widetilde{p}(\chi|\mathbf{x})\, d\chi \\
&= \int_{-\infty}^{\infty} \widetilde{p}(\chi|\mathbf{x})\, d\chi = 1
\end{aligned} \tag{3}
$$

and so the corresponding periodic density $p(\theta|\mathbf{x})$ will also be normalized. We now model the density function $\widetilde{p}(\chi|\mathbf{x})$ using a mixture of Gaussians of the form

$$\widetilde{p}(\chi|\mathbf{x}) = \sum_{i=1}^{m} \alpha_i(\mathbf{x})\phi_i(\chi|\mathbf{x}) \tag{4}$$

where the kernel functions are given by

$$\phi_i(\chi|\mathbf{x}) = \frac{1}{(2\pi)^{1/2}\sigma_i(\mathbf{x})} \exp\left(-\frac{\{\chi - \chi_i(\mathbf{x})\}^2}{2\sigma_i^2(\mathbf{x})}\right) \tag{5}$$

and the parameters $\alpha_i(\mathbf{x})$, $\sigma_i(\mathbf{x})$ and $\chi_i(\mathbf{x})$ are determined by the outputs of a feed-forward network. In particular, the mixing coefficients $\alpha_i(\mathbf{x})$ are governed by a 'softmax' activation function to ensure that they lie in the range $(0, 1)$ and sum to

unity; the width parameters $\sigma_i(\mathbf{x})$ are given by the exponentials of the corresponding network outputs to ensure their positivity; and the basis function centres $\chi_i(\mathbf{x})$ are given directly by network output variables.

The network is trained by maximizing the likelihood function, evaluated for set of training data, with respect to the weights and biases in the network. For a training set consisting of $N$ input vectors $\mathbf{x}^n$ and corresponding targets $\theta^n$, the likelihood is given by

$$\mathcal{L} = \prod_{n=1}^{N} p(\theta^n|\mathbf{x}^n)p(\mathbf{x}^n) \tag{6}$$

where $p(\mathbf{x})$ is the unconditional density of the input data. Rather than work with $\mathcal{L}$ directly, it is convenient instead to minimize an error function given by the negative log of the likelihood. Making use of (2) we can write this in the form

$$E = -\ln \mathcal{L} \simeq -\sum_{n} \ln \sum_{L} \tilde{p}(\theta^n + L2\pi|\mathbf{x}^n) \tag{7}$$

where we have dropped the term arising from $p(\mathbf{x})$ since it is independent of the network weights. This expression is very similar to the one which arises if we perform density estimation on the real axis, except for the extra summation over $L$, which means that the data point $\theta^n$ recurs at intervals of $2\pi$ along the $\chi$-axis. This is not equivalent simply to replicating the data, however, since the summation over $L$ occurs inside the logarithm, rather than outside as with the summation over data points $n$.

In a practical implementation, it is necessary to restrict the summation over $L$. For the results presented in the next section, this summation was taken over 7 complete periods of $2\pi$ spanning the range $(-7\pi, 7\pi)$. Since the Gaussians have exponentially decaying tails, this represents an extremely good approximation in almost all cases, provided we take care in initializing the network weights so that the Gaussian kernels lie in the central few periods. Derivatives of $E$ with respect to the network weights can be computed using the rules of calculus, to give a modified form of back-propagation. These derivatives can then be used with standard optimization techniques to find a minimum of the error function. (The results presented in the next section were obtained using the BFGS quasi-Newton algorithm).

## 2.2   MIXTURES OF CIRCULAR NORMAL DENSITIES

The second approach which we introduce is also based on a mixture of kernel functions of the form (1), but in this case the kernel functions themselves are periodic, thereby ensuring that the overall density function will be periodic. To motivate this approach, consider the problem of modelling the distribution of a velocity vector $\mathbf{v}$ in two dimensions (this arises, for example, in the application considered in Section 3). Since $\mathbf{v}$ lives in a Euclidean plane, we can model the density function $p(\mathbf{v})$ using a mixture of conventional spherical Gaussian kernels, where each kernel has

the form

$$\phi(v_x, v_y) = \frac{1}{2\pi\sigma^2} \exp\left(-\frac{\{v_x - \mu_x\}^2}{2\sigma^2} - \frac{\{v_y - \mu_y\}^2}{2\sigma^2}\right) \tag{8}$$

where $(v_x, v_y)$ are the Cartesian components of $\mathbf{v}$, and $(\mu_x, \mu_y)$ are the components of the center $\boldsymbol{\mu}$ of the kernel. From this we can extract the conditional distribution of the polar angle $\theta$ of the vector $\mathbf{v}$, given a value for $v = \|\mathbf{v}\|$. This is easily done with the transformation $v_x = v\cos\theta$, $v_y = v\sin\theta$, and defining $\theta_0$ to be the polar angle of $\boldsymbol{\mu}$, so that $\mu_x = \mu\cos\theta_0$ and $\mu_y = \mu\sin\theta_0$, where $\mu = \|\boldsymbol{\mu}\|$. This leads to a distribution which can be written in the form

$$\phi(\theta) = \frac{1}{2\pi I_0(\lambda)} \exp\left\{\lambda\cos(\theta - \theta_0)\right\} \tag{9}$$

where the normalization coefficient has been expressed in terms of the zeroth order modified Bessel function of the first kind, $I_0(\lambda)$. The distribution (9) is known as a *circular normal* or *von Mises* distribution (Mardia, 1972). The parameter $\lambda$ (which depends on $v$ in our derivation) is analogous to the (inverse) variance parameter in a conventional normal distribution. Since (9) is periodic, we can construct a general representation for the conditional density of a periodic variable by considering a mixture of circular normal kernels, with parameters given by the outputs of a neural network. The weights of the network can again be determined by maximizing the likelihood function defined over a set of training data.

## 2.3 FIXED KERNELS

The third approach introduced here is again based on a mixture model in which the kernel functions are periodic, but where the kernel parameters (specifying their width and location) are fixed. The only adaptive parameters are the mixing coefficients, which are again determined by the outputs of a feed-forward network having a softmax final-layer activation function. Here we consider a set of equally-spaced circular normal kernels in which the width parameters are chosen to give a moderate degree of overlap between the kernels so that the resulting representation for the density function will be reasonably smooth. Again, a maximum likelihood formalism is employed to train the network. Clearly a major drawback of fixed-kernel methods is that the number of kernels must grow exponentially with the dimensionality of the output space. For a single output variable, however, they can be regarded as practical techniques.

## 3 RESULTS

In order to test and compare the methods introduced above, we first consider a simple problem involving synthetic data, for which the true underlying distribution function is known. This data set is intended to mimic the central properties of the real data to be discussed in the next section. It has a single input variable $x$ and an output variable $\theta$ which lies in the range $(0, 2\pi)$. The distribution of $\theta$ is governed

by a mixture of two triangular functions whose parameters (locations and widths) are functions of $x$. Here we present preliminary results from the application of the method introduced in section 2.1 (involving the transformation to Euclidean space) to this data. Figure 2 shows a plot of the reconstructed conditional density in both the extended $\chi$ variable, and in the reconstructed polar variable $\theta$, for a particular value of the input variable $x$.

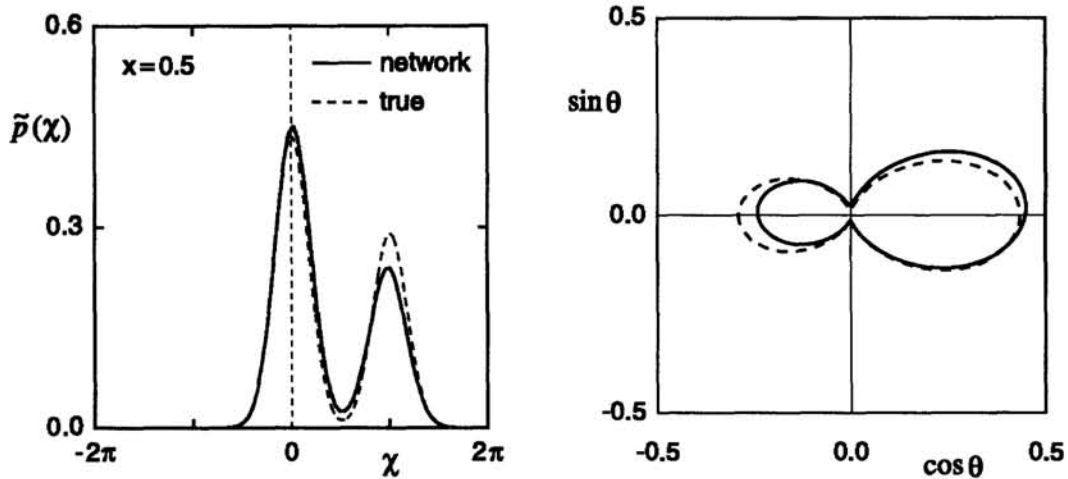

Figure 2: The left hand plot shows the predicted density (solid curve) together with the true density (dashed curve) in the extended $\chi$ space. The right hand plot shows the corresponding densities in the periodic $\theta$ space. In both cases the input variable is fixed at $x = 0.5$.

One of the original motivations for developing the techniques described in this paper was to provide an effective, principled approach to the analysis of radar scatterometer data from satellites such as the European Remote Sensing Satellite ERS-1. This satellite is equipped with three C-band radar antennae which measure the total backscattered power (called $\sigma_0$) along three directions relative to the satellite track, as shown in Figure 3. When the satellite passes over the ocean, the strengths of the backscattered signals are related to the surface ripples of the water (on length-scales of a few cm.) which in turn are determined by the low level winds. Extraction of the wind speed and direction from the radar signals represents an inverse problem which is typically multi-valued. For example, a wind direction of $\theta_1$ will give rise to similar radar signals to a wind direction of $\theta_1 + \pi$. Often, there are additional such 'aliases' at other angles. A conventional neural network approach to this problem, based on least-squares, would predict wind directions which were given by conditional averages of the target data. Since the average of several valid wind directions is typically not itself a valid direction, such an approach would clearly fail. Here we aim to extract the complete distribution of wind directions (as a function of the three $\sigma_0$ values and on the angle of incidence of the radar beam) and hence avoid

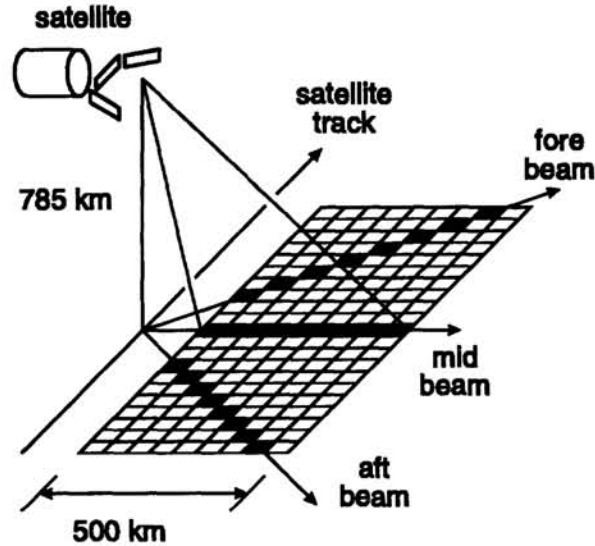

Figure 3: Schematic illustration of the ERS-1 satellite showing the footprints of the three radar scatterometers.

such difficulties. This approach also provides the most complete information for the next stage of processing (not considered here) which is to 'de-alias' the wind directions to extract the most probable overall wind field.

A large data set of ERS-1 measurements, spanning a wide range of meteorological conditions, has been assembled by the European Space Agency in collaboration with the UK Meteorological Office. Labelling of the data set was performed using wind vectors from the Meteorological Office Numerical Weather Prediction code. An example of the results from the fixed-kernel method of Section 2.3 are presented in Figure 4. This clearly shows the existence of a primary alias at an angle of $\pi$ relative to the principal direction, as well as secondary aliases at $\pm\pi/2$.

## Acknowledgements

We are grateful to the European Space Agency and the UK Meteorological Office for making available the ERS-1 data. We would also like to thank Iain Strachan and Ian Kirk of AEA Technology for a number of useful discussions relating to the interpretation of this data.

## References

Bishop C M (1994). Mixture density networks. Neural Computing Research Group Report, NCRG/4288, Department of Computer Science, Aston University, Birmingham, U.K.

Jacobs R A, Jordan M I, Nowlan S J and Hinton G E (1991). Adaptive mixtures

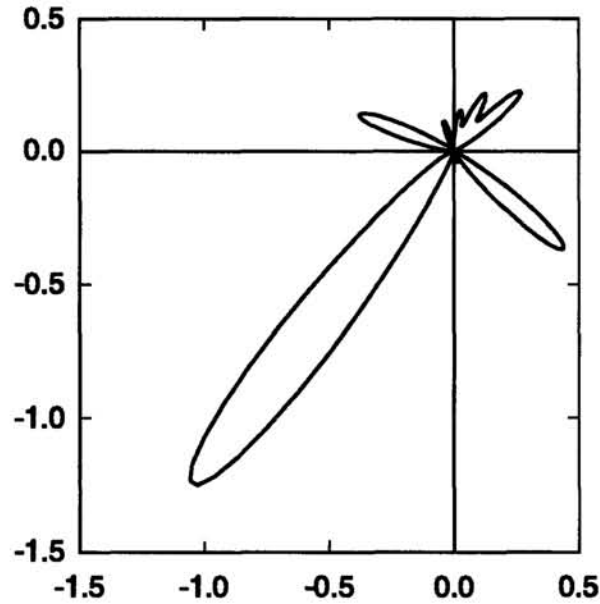

Figure 4: An example of the results obtained with the fixed-kernel method applied to data from the ERS-1 satellite. As well as the primary wind direction, there are aliases at $\pi$ and $\pm\pi/2$.

of local experts. *Neural Computation*, **3** 79–87.

Lui Y (1994) Robust parameter estimation and model selection for neural network regression. *Advances in Neural Information Processing Systems* **6** Morgan Kaufmann, 192–199..

Mardia K V (1972). *Statistics of Directional Data*. Academic Press, London.

White H (1992). Parametric statistical estimation with artificial neural networks. University of California, San Diego, Technical Report.
